# Predictive Matrix-Variate $t$ Models

**Shenghuo Zhu**    **Kai Yu**    **Yihong Gong**
NEC Labs America, Inc.
10080 N. Wolfe Rd. SW3-350
Cupertino, CA 95014
{zsh,kyu,ygong}@sv.nec-labs.com

## Abstract

It is becoming increasingly important to learn from a partially-observed random matrix and predict its missing elements. We assume that the entire matrix is a *single sample* drawn from a matrix-variate $t$ distribution and suggest a *matrix-variate $t$ model* (MVTM) to predict those missing elements. We show that MVTM generalizes a range of known probabilistic models, and automatically performs model selection to encourage sparse predictive models. Due to the non-conjugacy of its prior, it is difficult to make predictions by computing the mode or mean of the posterior distribution. We suggest an optimization method that sequentially minimizes a convex upper-bound of the log-likelihood, which is very efficient and scalable. The experiments on a toy data and EachMovie dataset show a good predictive accuracy of the model.

## 1   Introduction

Matrix analysis techniques, e.g., singular value decomposition (SVD), have been widely used in various data analysis applications. An important class of applications is to predict missing elements given a partially observed random matrix. For example, putting ratings of users into a matrix form, the goal of collaborative filtering is to predict those unseen ratings in the matrix.

To predict unobserved elements in matrices, the structures of the matrices play an importance role, for example, the similarity between columns and between rows. Such structures imply that elements in a random matrix are no longer independent and identically-distributed (*i.i.d.*). Without the *i.i.d.* assumption, many machine learning models are not applicable.

In this paper, we model the random matrix of interest as a single sample drawn from a matrix-variate $t$ distribution, which is a generalization of Student-$t$ distribution. We call the predictive model under such a prior by *matrix-variate $t$ model* (MVTM). Our study shows several interesting properties of the model. First, it continues the line of gradual generalizations across several known probabilistic models on random matrices, namely, from probabilistic principle component analysis (PPCA) [11], to Gaussian process latent-variable models (GPLVMs)[7], and to multi-task Gaussian processes (MTGPs) [13]. MVTMs can be further derived by analytically marginalizing out the hyper-parameters of these models. From a Bayesian modeling point of view, the marginalization of hyper-parameters means an automatic model selection and usually leads to a better generalization performance [8]; Second, the model selection by MVTMs explicitly encourages simpler predictive models that have lower ranks. Unlike the direct rank minimization, the log-determinant terms in the form of matrix-variate $t$ prior offers a continuous optimization surface (though non-convex) for rank constraint; Third, like multivariate Gaussian distributions, a matrix-variate $t$ prior is consistent under marginalization, that means, if a matrix follows a matrix-variate $t$ distribution, its any sub-matrix follows a matrix-variate $t$ distribution as well. This property allows to generalize distributions for finite matrices to infinite stochastic processes.

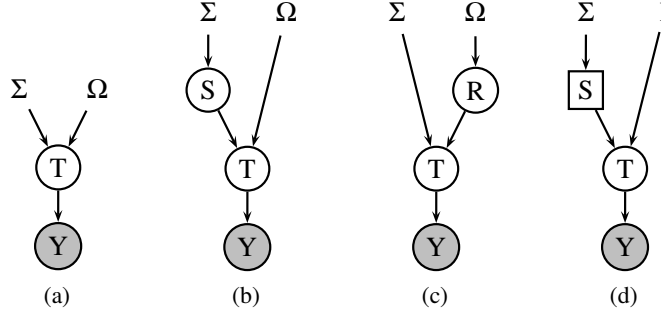

Figure 1: Models for matrix prediction. (a) MVTM. (b) and (c) are two normal-inverse-Wishart models, equivalent to MVTM when the covariance variable **S** (or **R**) is *marginalized*. (d) MTGP, which requires to *optimize* the covariance variable **S**. Circle nodes represent for random variables, shaded nodes for (partially) observable variables, text nodes for given parameters.

Under a Gaussian noise model, the matrix-variate $t$ distribution is not a conjugate prior. It is thus difficult to make predictions by computing the mode or mean of the posterior distribution. We suggest an optimization method that sequentially minimizes a convex upper-bound of the log-likelihood, which is highly efficient and scalable. In the experiments, the algorithm shows very good efficiency and excellent prediction accuracy.

This paper is organized as follows. We review three existing models and introduce the matrix-variate $t$ models in Section 2. The prediction methods are proposed in Section 3. In Section 4, the MVTM is compared with some other models. We illustrate the MVTM with the experiments on a toy example and on the movie-rating data in Section 5. We conclude in Section 6.

## 2 Predictive Matrix-Variate $t$ Models

### 2.1 A Family of Probabilistic Models for Matrix Data

In this section we introduce three probabilistic models in the literature. Let **Y** be a $p \times m$ observational matrix and **T** be the underlying $p \times m$ noise-free random matrix. We assume $\mathbf{Y}_{i,j} = \mathbf{T}_{i,j} + \epsilon_{i,j}$, $\epsilon_{i,j} \sim \mathcal{N}(0, \sigma^2)$, where $\mathbf{Y}_{i,j}$ denotes the $(i,j)$-th element of **Y**. If **Y** is partially observed, then $\mathbf{Y}_{\mathbb{I}}$ denotes the set of observed elements and $\mathbb{I}$ is the corresponding index set.

**Probabilistic Principal Component Analysis (PPCA)** [11] assumes that $\mathbf{y}_j$, the $j$-th column vector of **Y**, can be generated from a latent vector $\mathbf{v}_j$ in a $k$-dimensional linear space ($k < p$). The model is defined as $\mathbf{y}_j = \mathbf{W}\mathbf{v}_j + \boldsymbol{\mu} + \boldsymbol{\epsilon}_j$ and $\mathbf{v}_j \sim \mathcal{N}_k(\mathbf{v}_j; \mathbf{0}, \mathbf{I}_k)$, where $\boldsymbol{\epsilon}_j \sim \mathcal{N}_p(\boldsymbol{\epsilon}_j; \mathbf{0}, \sigma^2 \mathbf{I}_p)$, and **W** is a $p \times k$ loading matrix. By integrating out $\mathbf{v}_j$, we obtain the marginal distribution $\mathbf{y}_j \sim \mathcal{N}_p(\mathbf{y}_j; \boldsymbol{\mu}, \mathbf{W}\mathbf{W}^\top + \sigma^2 \mathbf{I}_p)$. Since the columns of **Y** are conditionally independent, letting **S** take the place of $\mathbf{W}\mathbf{W}^\top$, PPCA is similar[1] to

$$\mathbf{Y}_{i,j} = \mathbf{T}_{i,j} + \epsilon_{i,j}, \qquad \mathbf{T} \sim \mathcal{N}_{p,m}(\mathbf{T}; \mathbf{0}, \mathbf{S}, \mathbf{I}_m),$$

where $\mathcal{N}_{p,m}(\cdot; \mathbf{0}, \mathbf{S}, \mathbf{I}_m)$ is a matrix-variate normal prior with zero mean, covariance **S** between rows, and identity covariance $\mathbf{I}_m$ between columns. PPCA aims to estimate the parameter **W** by maximum likelihood.

**Gaussian Process Latent-Variable Model (GPLVM)** [7] formulates a latent-variable model in a slightly unconventional way. It considers the same linear relationship from latent representation $\mathbf{v}_j$ to observations $\mathbf{y}_j$. Instead of treating $\mathbf{v}_j$ as random variables, GPLVM assigns a prior on **W** and see $\{\mathbf{v}_j\}$ as parameters $\mathbf{y}_j = \mathbf{W}\mathbf{v}_j + \boldsymbol{\epsilon}_j$, and $\mathbf{W} \sim \mathcal{N}_{p,k}(\mathbf{W}; \mathbf{0}, \mathbf{I}_p, \mathbf{I}_k)$, where the elements of **W** are independent Gaussian random variables. By marginalizing out **W**, we obtain a distribution that each row of **Y** is an i.i.d. sample from a Gaussian process prior with the covariance $\mathbf{V}\mathbf{V}^\top + \sigma^2 \mathbf{I}_m$ and $\mathbf{V} = [\mathbf{v}_1, \ldots, \mathbf{v}_m]^\top$. Letting **R** take the place of $\mathbf{V}\mathbf{V}^\top$, we rewrite a similar model as

$$\mathbf{Y}_{i,j} = \mathbf{T}_{i,j} + \epsilon_{i,j}, \qquad \mathbf{T} \sim \mathcal{N}_{p,m}(\mathbf{T}; \mathbf{0}, \mathbf{I}_p, \mathbf{R}).$$

From a matrix modeling point of view, GPLVM estimates the covariance between the rows and assume the columns to be conditionally independent.

**Multi-task Gaussian Process** (MTGP) [13] is a multi-task learning model where each column of $\mathbf{Y}$ is a predictive function of one task, sampled from a Gaussian process prior, $\mathbf{y}_j = \mathbf{t}_j + \boldsymbol{\epsilon}_j$, and $\mathbf{t}_j \sim \mathcal{N}_p(\mathbf{0}, \mathbf{S})$, where $\boldsymbol{\epsilon}_j \sim \mathcal{N}_p(\mathbf{0}, \sigma^2 \mathbf{I}_p)$. It introduces a hierarchical model where an inverse-Wishart prior is added for the covariance,

$$\mathbf{Y}_{i,j} = \mathbf{T}_{i,j} + \epsilon_{i,j}, \qquad \mathbf{T} \sim \mathcal{N}_{p,m}(\mathbf{T}; \mathbf{0}, \mathbf{S}, \mathbf{I}_m), \qquad \mathbf{S} \sim \mathcal{IW}_p(\mathbf{S}; \nu, \mathbf{I}_p)$$

MTGP utilizes the inverse-Wishart prior as the regularization and obtains a maximum *a posteriori* (MAP) estimate of $\mathbf{S}$.

## 2.2 Matrix-Variate $t$ Models

The models introduced in the previous section are closely related to each other. PPCA models the row covariance of $\mathbf{Y}$, GPLVM models the column covariance, and MTGP assigns a hyper prior to prevent over-fitting when estimating the (row) covariance. From a matrix modeling point of view, capturing the dependence structure of $\mathbf{Y}$ by its row or column covariance is a matter of choices, which are not fundamentally different.[2] There is no reason to favor one choice over the other. By introducing the matrix-variate $t$ models (MVTMs), they can be unified to be the same model.

From a Bayesian modeling viewpoint, one should marginalize out as many variables as possible [8]. We thus extend the MTGP model in two directions: (1) assume $\mathbf{T} \sim \mathcal{N}_{p,m}(\mathbf{T}; \mathbf{0}, \mathbf{S}, \mathbf{I}_m)$ that have covariances on both sides of the matrix; (2) marginalize the covariance $\mathbf{S}$ on one side (see Figure 1(b)). Then we have a marginal distribution of $\mathbf{T}$

$$\Pr(\mathbf{T}) = \int \mathcal{N}_{p,m}(\mathbf{T}; \mathbf{0}, \mathbf{S}, \mathbf{I}_m)\mathcal{IW}_p(\mathbf{S}; \nu, \mathbf{I}_p)d\mathbf{S} = t_{p,m}(\mathbf{T}; \nu, \mathbf{0}, \mathbf{I}_p, \mathbf{I}_m), \qquad (1)$$

which is a matrix-variate $t$ distribution. Because the inverse-Wishart distribution may have different degree-of-freedom definition in literature, we use the definition in [5].

Following the definition in [6], the matrix-variate $t$ distribution of $p \times m$ matrix $\mathbf{T}$ is given by

$$t_{p,m}(\mathbf{T}; \nu, \mathbf{M}, \boldsymbol{\Sigma}, \boldsymbol{\Omega}) \stackrel{\text{def}}{=} \frac{1}{Z} |\boldsymbol{\Sigma}|^{-\frac{m}{2}} |\boldsymbol{\Omega}|^{-\frac{p}{2}} \left| \mathbf{I}_p + \boldsymbol{\Sigma}^{-1}(\mathbf{T} - \mathbf{M})\boldsymbol{\Omega}^{-1}(\mathbf{T} - \mathbf{M})^{\top} \right|^{-\frac{\nu+m+p-1}{2}},$$

where $\nu$ is the degree of freedom; $\mathbf{M}$ is a $p \times m$ matrix; $\boldsymbol{\Sigma}$ and $\boldsymbol{\Omega}$ are positive definite matrices of size $p \times p$ and $m \times m$, respectively; $Z = (\nu\pi)^{\frac{mp}{2}} \Gamma_p(\frac{\nu+p-1}{2})/\Gamma_p(\frac{\nu+m+p-1}{2})$; $\Gamma_p(\cdot)$ is a multivariate gamma function, and $|\cdot|$ stands for determinant.

The model can be depicted as Figure 1(a). One important property of matrix-variate $t$ distribution is that the marginal distribution of its sub-matrix still follows a matrix-variate $t$ distribution with the same degree of freedom (see Section 3.1). Therefore, we can expand it to the infinite dimensional stochastic process. By Eq. (1), we can see that Figure 1(a) and Figure 1(b) describe two equivalent models. Comparing them with the MTGP model represented in Figure 1(d), we can see that the difference lies in whether $\mathbf{S}$ is point estimated or integrated out.

Interestingly, the same matrix-variate $t$ distribution can be equivalently derived by putting another hierarchical generative process on the covariance $\mathbf{R}$, as described in Figure 1(c), where $\mathbf{R}$ follows an inverse-Wishart distribution. In other words, integrating the covariance on either side, we obtain the same model. This implies that the model controls the complexity of the covariances on both sides of the matrix. Neither PPCA nor GPLVM has such a property.

The matrix-variate $t$ distribution involves a determinant term of $\mathbf{T}$, which becomes a log-determinant term in log-likelihood or KL-divergence. The log-determinant term encourages the sparsity of matrix $\mathbf{T}$ with lower rank. This property has been used as the heuristic for minimizing the rank of the matrix in [3]. Student's $t$ priors were applied to enforce sparse kernel machine [10].

Here we say a few words about the given parameters. Though we can use evidence framework[8] or other methods to estimate $\nu$, the results are not good in many cases(see [4]). Usually we just set

$\nu$ to a small number. Similar to $\nu$, the estimated $\sigma^2$ does not give us a good result either, but cross-validation is a good choice. For the mean matrix $\mathbf{M}$, in our experiments, we just use sample average for all observed elements. For some tasks, when we have prior knowledge about the covariance between columns or between rows, we can use the covariance matrices in the places of $\mathbf{I}_m$ or $\mathbf{I}_p$.

## 3 Prediction Methods

When the evaluation of the prediction is the sum of individual losses, the optimal prediction is to find the individual mode of the marginal posterior distribution, i.e., $\arg\max_{\mathbf{T}_{ij}} \Pr(\mathbf{T}_{ij}|\mathbf{Y}_{\mathbb{I}})$. However, there is no exact solution for the marginal posterior. We have two ways to approximate the optimal prediction.

One way to make prediction is to compute the *mode* of the joint posterior distribution of $\mathbf{T}$, i.e. the prediction problem is

$$\widehat{\mathbf{T}} = \arg\max_{\mathbf{T}} \left\{\ln \Pr(\mathbf{Y}_{\mathbb{I}}|\mathbf{T}) + \ln \Pr(\mathbf{T})\right\}. \tag{2}$$

The computation of this estimation is usually easy. We discuss it in Section 3.3.

An alternative way is to use the individual *mean* of the posterior distribution to approximate the individual mode. Since the joint of individual mean happens to be the mean of the joint distribution, we only need to compute the joint posterior distribution. The problem of prediction by means is written as

$$\overline{\mathbf{T}} = \mathbb{E}(\mathbf{T}|\mathbf{Y}_{\mathbb{I}}). \tag{3}$$

However, it is usually difficult to compute the exact mean. One estimation method is the Monte Carlo method, which is computationally intensive. In Section 3.4, we discuss an approximation to compute the mean. From our experiments, the prediction by means usually outperforms the prediction by modes.

Before discussing the prediction methods, we introduce a few useful properties in Section 3.1 and suggest an optimization method as the efficient tool for prediction in Section 3.2.

### 3.1 Properties

The MVTM has a rich set of properties. We list a few in the following Theorem.

**Theorem 1.** *If*

$$
\begin{array}{cc} & \hspace{-1em}\begin{matrix} n & m \end{matrix} \\ \begin{matrix} q \\ p \end{matrix} & \hspace{-1em}\begin{pmatrix} \boldsymbol{\Theta} & \boldsymbol{\Phi} \\ \boldsymbol{\Psi} & \mathbf{T} \end{pmatrix} \end{array} \sim t_{p+q,m+n}\left(\cdot; \nu, \mathbf{0}, \begin{array}{cc} & \hspace{-1em}\begin{matrix} q & p \end{matrix} \\ \hspace{-1em}\begin{pmatrix} \mathbf{I}_q & \mathbf{0} \\ \mathbf{0} & \mathbf{I}_p \end{pmatrix} \end{array}, \begin{array}{cc} & \hspace{-1em}\begin{matrix} n & m \end{matrix} \\ \hspace{-1em}\begin{pmatrix} \mathbf{I}_n & \mathbf{0} \\ \mathbf{0} & \mathbf{I}_m \end{pmatrix} \end{array}\right), \tag{4}
$$

*then*

$$\Pr(\mathbf{T}) = t_{p,m}(\mathbf{T}; \nu, \mathbf{0}, \mathbf{I}_p, \mathbf{I}_m), \tag{5}$$

$$\Pr(\mathbf{T}|\boldsymbol{\Theta}, \boldsymbol{\Phi}, \boldsymbol{\Psi}) = t_{p,m}(\mathbf{T}; \nu + q + n, \mathbf{M}, (\mathbf{I}_p + \boldsymbol{\Psi}\mathbf{B}\boldsymbol{\Psi}^\top), (\mathbf{I}_m + \boldsymbol{\Phi}^\top\mathbf{A}\boldsymbol{\Phi})), \tag{6}$$

$$\Pr(\boldsymbol{\Theta}) = t_{q,n}(\boldsymbol{\Theta}; \nu, \mathbf{0}, \mathbf{I}_q, \mathbf{I}_n), \tag{7}$$

$$\Pr(\boldsymbol{\Phi}|\boldsymbol{\Theta}) = t_{q,m}(\boldsymbol{\Phi}; \nu + n, \mathbf{0}, \mathbf{A}^{-1}, \mathbf{I}_m), \tag{8}$$

$$\Pr(\boldsymbol{\Psi}|\boldsymbol{\Theta}, \boldsymbol{\Phi}) = t_{p,n}(\boldsymbol{\Psi}; \nu + q, \mathbf{0}, \mathbf{I}_p, \mathbf{B}^{-1}) = \Pr(\boldsymbol{\Psi}|\boldsymbol{\Theta}), \tag{9}$$

$$\mathbb{E}(\mathbf{T}|\boldsymbol{\Theta}, \boldsymbol{\Phi}, \boldsymbol{\Psi}) = \mathbf{M}, \tag{10}$$

$$\mathrm{Cov}\left(\mathrm{vec}\left(\mathbf{T}^\top\right)|\boldsymbol{\Theta}, \boldsymbol{\Phi}, \boldsymbol{\Psi}\right) = (\nu + q + n - 2)^{-1}(\mathbf{I}_p + \boldsymbol{\Psi}\mathbf{B}\boldsymbol{\Psi}^\top) \otimes (\mathbf{I}_m + \boldsymbol{\Phi}^\top\mathbf{A}\boldsymbol{\Phi}), \tag{11}$$

*where* $\mathbf{A} \stackrel{def}{=} (\boldsymbol{\Theta}\boldsymbol{\Theta}^\top + \mathbf{I}_q)^{-1}$, $\mathbf{B} \stackrel{def}{=} (\boldsymbol{\Theta}^\top\boldsymbol{\Theta} + \mathbf{I}_n)^{-1}$, *and* $\mathbf{M} \stackrel{def}{=} \boldsymbol{\Psi}\boldsymbol{\Theta}^\top\mathbf{A}\boldsymbol{\Phi} = \boldsymbol{\Psi}\mathbf{B}\boldsymbol{\Theta}^\top\boldsymbol{\Phi}$.

This theorem can be directly derived from Theorem 4.3.1 and 4.3.9 in [6] with a little calculus. It provides some insights about MVTMs. The marginal distribution in Eq. (5) has the same form as the joint distribution, therefore the matrix-variate $t$ distribution is *extensible* to an infinite dimensional stochastic process. As conditional distribution in Eq. (6) is still a matrix-variate $t$ distribution, we can use it to approximate the posterior distribution, which we use in Section 3.4.

We encounter log-determinant terms in computation of the mode or mean estimation. The following theorem provides a quadratic upper bounds for the log-determinant terms, which makes it possible to apply the optimization method in Section 3.2.

**Lemma 1.** *If $\mathbf{X}$ is a $p \times p$ positive definite matrices, it holds that $\ln|\mathbf{X}| \leq \text{tr}(\mathbf{X}) - p$. The equality holds when $\mathbf{X}$ is an orthonormal matrix.*

*Proof.* Let $\{\lambda_1, \cdots, \lambda_p\}$ be the eigenvalues of $\mathbf{X}$. We have $\ln|\mathbf{X}| = \sum_i \ln \lambda_i$ and $\text{tr}(\mathbf{X}) = \sum_i \lambda_i$. Since $\ln \lambda_i \leq \lambda_i - 1$, we have the inequality. The equality holds when $\lambda_i = 1$. Therefore, when $\mathbf{X}$ is an orthonormal matrix (especially $\mathbf{X} = \mathbf{I}_p$), the equality holds. $\qquad\square$

**Theorem 2.** *If $\boldsymbol{\Sigma}$ is a $p \times p$ positive definite matrix, $\boldsymbol{\Omega}$ is an $m \times m$ positive definite matrix, and $\mathbf{T}$ and $\mathbf{T}_0$ are $p \times m$ matrices, it holds that*

$$\ln|\boldsymbol{\Sigma} + \mathbf{T}\boldsymbol{\Omega}^{-1}\mathbf{T}^{\top}| \leq h(\mathbf{T}; \mathbf{T}_0, \boldsymbol{\Sigma}, \boldsymbol{\Omega}) + h_0(\mathbf{T}_0, \boldsymbol{\Sigma}, \boldsymbol{\Omega}),$$

*where*

$$h(\mathbf{T}; \mathbf{T}_0, \boldsymbol{\Sigma}, \boldsymbol{\Omega}) \overset{def}{=} \text{tr}\left((\boldsymbol{\Sigma} + \mathbf{T}_0\boldsymbol{\Omega}^{-1}\mathbf{T}_0^{\top})^{-1}\mathbf{T}\boldsymbol{\Omega}^{-1}\mathbf{T}^{\top}\right),$$

$$h_0(\mathbf{T}_0, \boldsymbol{\Sigma}, \boldsymbol{\Omega}) \overset{def}{=} \ln|\boldsymbol{\Sigma} + \mathbf{T}_0\boldsymbol{\Omega}^{-1}\mathbf{T}_0^{\top}| + \text{tr}\left((\boldsymbol{\Sigma} + \mathbf{T}_0\boldsymbol{\Omega}^{-1}\mathbf{T}_0^{\top})^{-1}\boldsymbol{\Sigma}\right) - p$$

*The equality holds when $\mathbf{T} = \mathbf{T}_0$. Also it holds that*

$$\left.\frac{\partial}{\partial \mathbf{T}}h(\mathbf{T}; \mathbf{T}_0, \boldsymbol{\Sigma}, \boldsymbol{\Omega})\right|_{\mathbf{T}=\mathbf{T}_0} = 2(\boldsymbol{\Sigma} + \mathbf{T}_0\boldsymbol{\Omega}^{-1}\mathbf{T}_0^{\top})^{-1}\mathbf{T}_0\boldsymbol{\Omega}^{-1} = \left.\frac{\partial}{\partial \mathbf{T}}\ln|\boldsymbol{\Sigma} + \mathbf{T}\boldsymbol{\Omega}^{-1}\mathbf{T}^{\top}|\right|_{\mathbf{T}=\mathbf{T}_0}.$$

Applying Lemma 1 with $\mathbf{X} = (\boldsymbol{\Sigma} + \mathbf{T}_0\boldsymbol{\Omega}^{-1}\mathbf{T}_0^{\top})^{-1}(\boldsymbol{\Sigma} + \mathbf{T}\boldsymbol{\Omega}^{-1}\mathbf{T}^{\top})$, we obtain the inequality. By some calculus we have the equality of the first-order derivative. Actually $h(\cdot)$ is a quadratic convex function with respect to $\mathbf{T}$, as $(\boldsymbol{\Sigma} + \mathbf{T}_0\boldsymbol{\Omega}^{-1}\mathbf{T}_0^{\top})^{-1}$ and $\boldsymbol{\Omega}^{-1}$ are positive definite matrices.

## 3.2 Optimization Method

Once the objective is given, the prediction becomes an optimization problem. We use an EM-style optimization method to make the prediction. Suppose $\mathcal{J}(\mathbf{T})$ be the objective function to be minimized. If we can find an auxiliary function, $\mathcal{Q}(\mathbf{T}; \mathbf{T}')$, having the following properties, we can apply this method.

1. $\mathcal{J}(\mathbf{T}) \leq \mathcal{Q}(\mathbf{T}; \mathbf{T}')$ and $\mathcal{J}(\mathbf{T}') = \mathcal{Q}(\mathbf{T}'; \mathbf{T}')$,
2. $\partial \mathcal{J}(\mathbf{T})/\partial \mathbf{T}|_{\mathbf{T}=\mathbf{T}'} = \partial \mathcal{Q}(\mathbf{T}; \mathbf{T}')/\partial \mathbf{T}|_{\mathbf{T}=\mathbf{T}'}$,
3. For a fixed $\mathbf{T}'$, $\mathcal{Q}(\mathbf{T}; \mathbf{T}')$ is quadratic and convex with respect to $\mathbf{T}$.

Starting from any $\mathbf{T}_0$, as long as we can find a $\mathbf{T}_1$ such that $\mathcal{Q}(\mathbf{T}_1, \mathbf{T}_0) < \mathcal{Q}(\mathbf{T}_0, \mathbf{T}_0)$, we have $\mathcal{J}(\mathbf{T}_0) = \mathcal{Q}(\mathbf{T}_0, \mathbf{T}_0) > \mathcal{Q}(\mathbf{T}_1, \mathbf{T}_0) \geq \mathcal{J}(\mathbf{T}_1)$. If there exists a global minimum point of $\mathcal{J}(\mathbf{T})$, there exists a global minimum point of $\mathcal{Q}(\mathbf{T}; \mathbf{T}_0)$ as well, because $\mathcal{Q}(\mathbf{T}; \mathbf{T}_0)$ is upper bound of $\mathcal{J}(\mathbf{T})$. Since $\mathcal{Q}(\mathbf{T}; \mathbf{T}_0)$ is quadratic with the respect to $\mathbf{T}$, we can apply the Newton-Raphson method to minimize $\mathcal{Q}(\mathbf{T}; \mathbf{T}_0)$. As long as $\mathbf{T}_0$ is not a local minimum, maximum or saddle point of $\mathcal{J}$, we can find a $\mathbf{T}$ to reduce $\mathcal{Q}(\mathbf{T}; \mathbf{T}_0)$, because $\mathcal{Q}(\mathbf{T}; \mathbf{T}_0)$ has the same derivative as $\mathcal{J}(\mathbf{T})$ at $\mathbf{T}_0$. Usually, a random starting point, $\mathbf{T}_0$ is unlikely to be a local maximum, then $\mathbf{T}_1$ can not be a local maximum. If $\mathbf{T}_0$ is a local maximum, we can reselect a point, which is not. After we find a $\mathbf{T}_i$, we repeat the procedure to find a $\mathbf{T}_{i+1}$ so that $\mathcal{J}(\mathbf{T}_{i+1}) < \mathcal{J}(\mathbf{T}_i)$, unless $\mathbf{T}_i$ is a local minimum or saddle point of $\mathcal{J}$. Repeating this procedure, $\mathbf{T}_i$ converges a local minimum or saddle point of $\mathcal{J}$, as long as $\mathbf{T}_0$ is not a local maximum.

## 3.3 Mode Prediction

Following Eq. (2), the goal is to minimize the objective function

$$\widehat{\mathcal{J}}(\mathbf{T}) \overset{def}{=} \ell(\mathbf{T}) + \frac{\nu+m+p-1}{2}\ln\left|\mathbf{I}_p + \mathbf{T}\mathbf{T}^{\top}\right|, \tag{12}$$

where $\ell(\mathbf{T}) \stackrel{\text{def}}{=} -\ln \Pr(\mathbf{Y}_{\mathbb{I}}) = \frac{1}{2\sigma^2} \sum_{(i,j)\in\mathbb{I}} (\mathbf{T}_{ij} - \mathbf{Y}_{ij})^2 + const.$

As $\widehat{\mathcal{J}}$ contains a log-determinant term, minimizing $\widehat{\mathcal{J}}$ by nonlinear optimization is slow. Here, we introduce an auxiliary function,

$$\mathcal{Q}(\mathbf{T}; \mathbf{T}') \stackrel{\text{def}}{=} \ell(\mathbf{T}) + h(\mathbf{T}; \mathbf{T}', \mathbf{I}_p, \mathbf{I}_m) + h_0(\mathbf{T}', \mathbf{I}_p, \mathbf{I}_m). \tag{13}$$

By Corollary 2, we have that $\widehat{\mathcal{J}}(\mathbf{T}) \leq \mathcal{Q}(\mathbf{T}; \mathbf{T}')$, $\widehat{\mathcal{J}}(\mathbf{T}') = \mathcal{Q}(\mathbf{T}', \mathbf{T}')$, and $\mathcal{Q}(\mathbf{T}, \mathbf{T}')$ has the same first-order derivative as $\widehat{\mathcal{J}}(\mathbf{T})$ at $\mathbf{T}'$. Because $l$ and $h$ are quadratic and convex, $\mathcal{Q}$ is quadratic and convex as well. Therefore, we can apply the optimization method in Section 3.2 to minimize $\widehat{\mathcal{J}}$.

However, when the size of $\mathbf{T}$ is large, to find $\widehat{\mathbf{T}}$ is still time consuming and requires a very large space. In many tasks, we only need to infer a small portion of $\widehat{\mathbf{T}}$. Therefore, we consider a low rank approximation, using $\mathbf{U}\mathbf{V}^\top$ to approximate $\mathbf{T}$, where $\mathbf{U}$ is a $p \times k$ matrix and $\mathbf{V}$ is an $m \times k$ matrix. The problem of Eq. (2) is approximated by $\arg\min_{\mathbf{U},\mathbf{V}} \widehat{\mathcal{J}}(\mathbf{U}\mathbf{V}^\top)$. We can minimize $\mathbf{J}_1$ by alternatively optimizing $\mathbf{U}$ and $\mathbf{V}$. We can put the final result in a canonical format as $\widehat{\mathbf{T}} \approx \mathbf{U}\mathbf{S}\mathbf{V}^\top$, where $\mathbf{U}$ and $\mathbf{V}$ are semi-orthonormal and $\mathbf{S}$ is a $k \times k$ diagonal matrix. This result can be consider as the SVD of an incomplete matrix using matrix-variate $t$ regularization. The details are skipped because of the limit space.

### 3.4 Variational Mean Prediction

As the difficulty in explicitly computing the posterior distribution of $\mathbf{T}$, we take a variational approach to approximate its posterior distribution by a matrix-variate $t$ distribution via an expanded model. We expand the model by adding matrix variate $\boldsymbol{\Theta}$, $\boldsymbol{\Phi}$ and $\boldsymbol{\Psi}$ with distribution as Eq. (4). Since the marginal distribution, Eq. (5), is the same as the prior of $\mathbf{T}$, we can derive the original model by marginalizing out $\boldsymbol{\Theta}$, $\boldsymbol{\Phi}$ and $\boldsymbol{\Psi}$. However, instead of integrating out $\boldsymbol{\Theta}$, $\boldsymbol{\Phi}$ and $\boldsymbol{\Psi}$, we use them as the parameters to approximate $\mathbf{T}$'s posterior distribution. Therefore, the estimation of the parameters is to minimize

$$-\ln \Pr(\mathbf{Y}_{\mathbb{I}}, \boldsymbol{\Theta}, \boldsymbol{\Phi}, \boldsymbol{\Psi}) = -\ln \Pr(\boldsymbol{\Theta}, \boldsymbol{\Phi}, \boldsymbol{\Psi}) - \ln \int \Pr(\mathbf{T}|\boldsymbol{\Theta}, \boldsymbol{\Phi}, \boldsymbol{\Psi}) \Pr(\mathbf{Y}_{\mathbb{I}}|\mathbf{T}) d\mathbf{T} \tag{14}$$

over $\boldsymbol{\Theta}$, $\boldsymbol{\Phi}$ and $\boldsymbol{\Psi}$. The first term in the RHS of Eq. (14) can be written as

$$\begin{aligned}
-\ln \Pr(\boldsymbol{\Theta}, \boldsymbol{\Phi}, \boldsymbol{\Psi}) &= -\ln \Pr(\boldsymbol{\Theta}) - \ln \Pr(\boldsymbol{\Phi}|\boldsymbol{\Theta}) - \ln \Pr(\boldsymbol{\Psi}|\boldsymbol{\Theta}, \boldsymbol{\Phi}) \\
&= \frac{\nu+q+n+p+m-1}{2} \ln |\mathbf{I}_q + \boldsymbol{\Theta}\boldsymbol{\Theta}^\top| + \frac{\nu+q+n+m-1}{2} \ln |\mathbf{I}_m + \boldsymbol{\Phi}^\top \mathbf{A}\boldsymbol{\Phi}| \\
&\quad + \frac{\nu+q+n+p-1}{2} \ln |\mathbf{I}_p + \boldsymbol{\Psi}\mathbf{B}\boldsymbol{\Psi}^\top| + const.
\end{aligned} \tag{15}$$

Due to the convexity of negative logarithm, the second term in the RHS of Eq. (14) is bounded by

$$\ell(\boldsymbol{\Psi}\mathbf{B}^{\frac{1}{2}}\boldsymbol{\Theta}^\top \mathbf{A}^{\frac{1}{2}}\boldsymbol{\Phi}) + \frac{1}{2\sigma^2(\nu+q+n-2)} \sum_{(i,j)\in\mathbb{I}} (1 + [\boldsymbol{\Psi}\mathbf{B}\boldsymbol{\Psi}^\top]_{ii})(1 + [\boldsymbol{\Phi}^\top \mathbf{A}\boldsymbol{\Phi}]_{jj}) + const. \tag{16}$$

because $-\ln \Pr(\mathbf{Y}_{\mathbb{I}}|\mathbf{T})$ is quadratic respective to $\mathbf{T}$, thus we only need integration using the mean and variance of $T_{ij}$ of $\Pr(\mathbf{T}|\boldsymbol{\Theta}, \boldsymbol{\Phi}, \boldsymbol{\Psi})$, which is given by Eq. (10) and (11). The parameter estimation not only reduce the loss (the term of $\ell(\cdot)$), but also reduce the variance. Because of this, the prediction by means usually outperforms the prediction by modes.

Let $\overline{\mathcal{J}}$ be the sum of the right-hand-side of Eq. (15) and (16), which can be considered as the upper bound of Eq. (14) (ignoring constants). Here, we estimate the parameters by minimizing $\overline{\mathcal{J}}$. Because $\mathbf{A}$ and $\mathbf{B}$ involve the inverse of quadratic term of $\boldsymbol{\Theta}$, it is awkward to directly optimize $\boldsymbol{\Theta}, \boldsymbol{\Phi}, \boldsymbol{\Psi}$. We reparameterize $\overline{\mathcal{J}}$ by $\mathbf{U} \stackrel{\text{def}}{=} \boldsymbol{\Psi}\mathbf{B}^{1/2}$, $\mathbf{V} \stackrel{\text{def}}{=} \boldsymbol{\Phi}^\top \mathbf{A}^{1/2}$, and $\mathbf{S} \stackrel{\text{def}}{=} \boldsymbol{\Theta}$. We can easily apply the optimization method in Section 3.2 to find optimal $\mathbf{U}$, $\mathbf{V}$ and $\mathbf{S}$. After estimation $\mathbf{U}$, $\mathbf{V}$ and $\mathbf{S}$, by Theorem 1, we can compute $\overline{\mathbf{T}} = \mathbf{M} = \mathbf{U}\mathbf{S}\mathbf{V}^\top$. The details are skipped because of the limit space.

## 4   Related work

**Maximum Margin Matrix Factorization** (MMMF) [9] is not in the framework of stochastic matrix analysis, but there are some similarities between MMMF and our mode estimation in Section 3.3.

Using trace norm on the matrix as regularization, MMMF overcomes the over-fitting problem in factorizing matrix with missing values. From the regularization viewpoint, the prediction by mode of MVTM uses log-determinants as the regularization term in Eq. (12). The log-determinants encourage sparsity predictive models.

**Stochastic Relational Models** (SRMs) [12] extend MTGPs by estimating the covariance matrices for each side. The covariance functions are required to be estimated from observation. By maximizing marginalized likelihood, the estimated $\mathbf{S}$ and $\mathbf{R}$ reflect the information of the dependency structure. Then the relationship can be predicted with $\mathbf{S}$ and $\mathbf{R}$. During estimating $\mathbf{S}$ and $\mathbf{R}$, inverse-Wishart priors with parameter $\boldsymbol{\Sigma}$ and $\boldsymbol{\Omega}$ are imposed to $\mathbf{S}$ and $\mathbf{R}$ respectively. MVTM differs from SRM in integrating out the hyper-parameters or maximizing out. As MacKay suggests [8], "one should integrate over as many variables as possible".

**Robust Probabilistic Projections** (RPP)[1] uses Student-$t$ distribution to extends PPCA by scaling each feature vector by an independent random variable. Written in a matrix format, RPP is

$$\mathbf{T} \sim \mathcal{N}_{p,m}(\mathbf{T}; \boldsymbol{\mu}\mathbf{1}^\top, \mathbf{W}\mathbf{W}^\top, \mathbf{U}), \qquad \mathbf{U} = \mathrm{diag}\{u_i\}, \qquad u_i \sim \mathcal{IG}(u_i|\frac{\nu}{2}, \frac{\nu}{2}),$$

where $\mathcal{IG}$ is inverse Gamma distribution. Though RPP unties the scale factors between feature vectors, which could make the estimation more robust, it does not integrate out the covariance matrix, which we did in MVTM. Moreover inherited from PPCA, RPP implicitly uses independence assumption of feature vectors. Also RPP results different models depending on which side we assume to be independent, therefore it is not suitable for matrix prediction.

# 5 Experiments

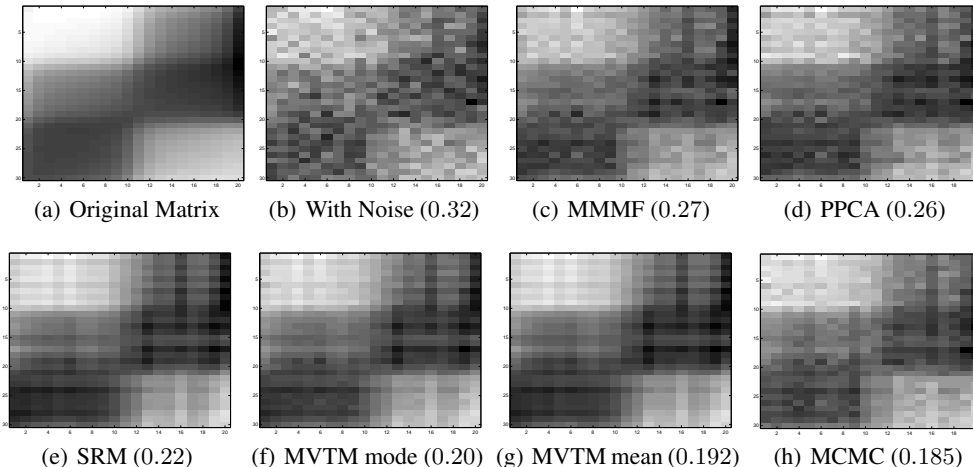

(a) Original Matrix     (b) With Noise (0.32)     (c) MMMF (0.27)     (d) PPCA (0.26)

(e) SRM (0.22)     (f) MVTM mode (0.20)     (g) MVTM mean (0.192)     (h) MCMC (0.185)

Figure 2: Experiments on synthetic data. RMSEs are shown in parentheses.

**Synthetic data:** We generate a $30 \times 20$ matrix (Figure 2(a)), then add noise with $\sigma^2 = 0.1$ (Figure 2(b)). The root mean squared noise is $0.32$. We select $70\%$ elements as the observed data and the rest elements are for prediction. We apply MMMF [9], PPCA[11], MTGP[13], SRM [12], our MVTM prediction-by-means and prediction-by-modes methods. The number of dimensions for low rank approximation is 10. We also apply MCMC method to infer the matrix. The reconstruction matrix and root mean squared errors of prediction on the unobserved elements (comparing to the original matrix) are shown in Figure 2(c)-2(g), respectively. MTGP has the similar result as PPCA, we do not show the result.

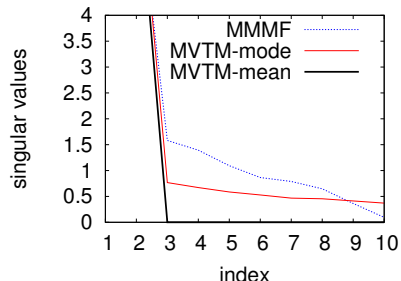

Figure 3: Singular values of recovered matrices in descent order.

MVTM is in favor of sparse predictive models. To verify this, we depict the singular values of the MMMF method and two MVTM prediction methods in Figure 3. There are only two singular

|      | user mean | movie mean | MMMF | PPCA | MVTM (mode) | MVTM (mean) |
|------|-----------|------------|------|------|-------------|-------------|
| RMSE | 1.425     | 1.387      | 1.186 | 1.165 | 1.162      | 1.151       |
| MAE  | 1.141     | 1.103      | 0.943 | 0.915 | 0.898      | 0.887       |

Table 1: RMSE (root mean squred error) and MAE (mean absolute error) of experiments on Each-movie data. All standard errors are 0.001 or less.

values of the MVTM prediction-by-means method are non-zeros. The singular values of the mode estimation decrease faster than the MMMF ones at beginning, but decrease slower after a threshold. This confirms that the log-determinants automatically determine the intrinsic rank of the matrices.

**Eachmovie data:** We test our algorithms on Eachmovie from [2]. The dataset contains $74,424$ users' $2,811,718$ ratings on $1,648$ movies, i.e. about $2.29\%$ are rated by zero-to-five stars. We put all ratings into a matrix, and randomly select $80\%$ as observed data to predict the remaining ratings. The random selection was carried out 10 times independently. We compare our approach with other three approaches: 1) USER MEAN predicting rating by the sample mean of the same user' ratings; 2) MOVIE MEAN, predicting rating by the sample mean of users' ratings of the same movie; 3) MMMF[9]; 4) PPCA[11]. We do not have a scalable implementation for other approaches compared in the previous experiment. The number of dimensions is 10. The results are shown in Table 1. Two MVTM prediction methods outperform the other methods.

## 6  Conclusions

In this paper we introduce matrix-variate $t$ models for matrix prediction. The entire matrix is modeled as a sample drawn from a matrix-variate $t$ distribution. An MVTM does not require the independence assumption over elements. The implicit model selection of the MVTM encourages sparse models with lower ranks. To minimize the log-likelihood with log-determinant terms, we propose an optimization method by sequentially minimizing its convex quadratic upper bound. The experiments show that the approach is accurate, efficient and scalable.

## Footnotes

[1]Because it requires **S** to be positive definite and **W** is usually low rank, they are not equivalent.

[2]GPLVM offers an advantage of using nonlinear covariance function based on attributes.

## References

[1] C. Archambeau, N. Delannay, and M. Verleysen. Robust probabilistic projections. In *ICML*, 2006.

[2] J. Breese, D. Heckerman, and C. Kadie. Empirical analysis of predictive algorithms for collaborative filtering. In *UAI-98*, pages 43–52, 1998.

[3] M. Fazel, H. Haitham, and S. P. Boyd. Log-det heuristic for matrix rank minimization with applications to hankel and euclidean distance matrices. In *Proceedings of the American Control Conference*, 2003.

[4] C. Fernandez and M. F. J. Steel. Multivariate Student-$t$ regression models: Pitfalls and inference. *Biometrika*, 86(1):153–167, 1999.

[5] A. Gelman, J. B. Carlin, H. S. Stern, and D. B. Rubin. *Bayesian Data Analysis*. Chapman & Hall/CRC, New York, 2nd edition, 2004.

[6] A. K. Gupta and D. K. Nagar. *Matrix Variate Distributions*. Chapman & Hall/CRC, 2000.

[7] N. Lawrence. Probabilistic non-linear principal component analysis with gaussian process latent variable models. *J. Mach. Learn. Res.*, 6:1783–1816, 2005.

[8] D. J. C. MacKay. Comparison of approximate methods for handling hyperparameters. *Neural Comput.*, 11(5):1035–1068, 1999.

[9] J. D. M. Rennie and N. Srebro. Fast maximum margin matrix factorization for collaborative prediction. In *ICML*, 2005.

[10] M. E. Tipping. Sparse bayesian learning and the relevance vector machine. *Journal of Machine Learning Research*, 1:211–244, 2001.

[11] M. E. Tipping and C. M. Bishop. Probabilistic principal component analysis. *Journal of the Royal Statisitical Scoiety*, B(61):611–622, 1999.

[12] K. Yu, W. Chu, S. Yu, V. Tresp, and Z. Xu. Stochastic relational models for discriminative link prediction. In *Advances in Neural Information Processing Systems 19 (NIPS)*, 2006.

[13] K. Yu, V. Tresp, and A. Schwaighofer. Learning Gaussian processes from multiple tasks. In *ICML*, 2005.
